# An Hierarchical Model of Visual Rivalry

Peter Dayan
Department of Brain and Cognitive Sciences
E25-210 Massachusetts Institute of Technology
Cambridge, MA 02139
dayan@psyche.mit.edu[1]

## Abstract

Binocular rivalry is the alternating percept that can result when the two eyes see different scenes. Recent psychophysical evidence supports an account for one component of binocular rivalry similar to that for other bistable percepts. We test the hypothesis[19,16,18] that alternation can be generated by competition between top-down cortical explanations for the inputs, rather than by direct competition between the inputs. Recent neurophysiological evidence shows that some binocular neurons are modulated with the changing percept; others are not, even if they are selective between the stimuli presented to the eyes. We extend our model to a hierarchy to address these effects.

## 1  Introduction

Although binocular rivalry leads to distinct perceptual distress, it is revealing about the mechanisms of visual information processing. The first accounts for rivalry argued on the basis of phenomena such as increases in thresholds for test stimuli presented in the suppressed eye[24,8,3] that there was a early competitive process, the outcome of which meant that the system would just ignore input from one eye in favour of the other. Various experiments have suggested that simple input competition cannot be the whole story. For instance, in a case in which rivalry is between a vertical grating in the left eye and a horizontal one in the right, and in which a vertical grating is presented prior to rivalry to cause adaptation, the relative suppression of vertical during rivalry is independent of

[1]I am very grateful to Bart Anderson, Adam Elga, Geoff Goodhill, Geoff Hinton, David Leopold, Earl Miller, Read Montague, Bruno Olshausen, Pawan Sinha, Rich Zemel, and particularly Zhaoping Li and Tommi Jaakkola for their comments on earlier drafts and discussions. This work was supported by the NIH.

the eye of origin of the adapting grating.[4] Even more compelling, if the rivalrous stimuli in the two eyes are switched rapidly, the percept switches only slowly – competition is more between coherent percepts than merely inputs. Rivalry is an attractive paradigm for studying models of cortex like the Helmholtz machine[12,7] that construct coherent percepts, and in particular for studying *hierarchical* models, because of electrophysiological data on the behaviour during rivalry of cells at different levels of the visual processing hierarchy.[16]

Leopold & Logothetis[16] trained monkeys to report their percepts during rivalrous and non-rivalrous stimuli whilst recording from neurons V1/2 and V4. Important findings are that striate monocular neurons are unaffected by rivalry; some striate binocular neurons that are selective between the stimuli modulate their activities during rivalry; others do not; some fire more when their preferred stimuli are suppressed; others still are only selective during rivalry. In this paper we consider one form of analysis-by-synthesis model of cortical processing[7] and show how it can exhibit rivalry between explanations in the case that the eyes receive different input. This model can provide an account for many of the behaviours described above.

## 2   The Model

Figure 1a shows the full generative model. Units in layers $\mathbf{y}$ (modeling V1) and $\mathbf{x}$ and $\mathbf{w}$ (modeling early and late extra-striate areas) are all binocular and jointly explain successively more complex features in the input $\mathbf{z}$ according to a top-down generative model. Apart from the half bars in $\mathbf{y}$, the model is similar to that learned by the Helmholtz machine[12,7] for which increasing complexity in higher layers rather than the increasing input scale is key. In this case, for instance, $w_2$ specifies the occurrence of vertical bars anywhere in the $8 \times 8$ input grids; $x_{16}$ specifies the rightmost vertical bar; and $y_{31}$ and $y_{32}$ the top and bottom half of this vertical bar. These specifications are provided by a top-down generative model in which the activations of units are specified by probabilities such as $\mathcal{P}[y_i = 1|\mathbf{x}] = \sigma\left(b_y + \sum_k x_k J_{\mathbf{xy}}^{ki}\right)$ where the sum $k$ is over all the units in the $\mathbf{x}$ layer, and $\sigma()$ is a robust normal distribution function. We model the percept in terms of the activation in the $\mathbf{w}$ layer.

We model differing input contrasts by representing the input to $z_i$ by $d_i$, where $\mathcal{P}[z_i = 1] = \sigma(d_i)$ and all the $z_i$ are independent. Recognition is formally the statistical inverse to generation, and should produce distribution $\mathcal{P}[\mathbf{w}, \mathbf{x}, \mathbf{y}|\mathbf{d}]$ over all the choices of the hidden activations. We use a mean field inversion method,[13] using a factorised approximation $\mathcal{Q}[\mathbf{w}, \mathbf{x}, \mathbf{y}; \mu, \xi, \psi] = \mathcal{Q}[\mathbf{w}; \mu]\mathcal{Q}[\mathbf{x}; \xi]\mathcal{Q}[\mathbf{y}; \psi]$, with $\mathcal{Q}[\mathbf{w}; \mu] = \prod_i \sigma(\mu_i)^{w_i}(1 - \sigma(\mu_i))^{1-w_i}$, *etc*, and fitting the parameters $\mu, \xi, \psi$ to minimise the approximation cost:

$$\mathcal{F}[\mu, \xi, \psi] = \sum_{\mathbf{z}} \mathcal{P}[\mathbf{z}; \mathbf{d}] \sum_{\mathbf{w}, \mathbf{x}, \mathbf{y}} \mathcal{Q}[\mathbf{w}, \mathbf{x}, \mathbf{y}; \mu, \xi, \psi] \log \frac{\mathcal{Q}[\mathbf{w}, \mathbf{x}, \mathbf{y}; \mu, \xi, \psi]}{\mathcal{P}[\mathbf{w}, \mathbf{x}, \mathbf{y}|\mathbf{z}]}.$$

We report the mean activities of the units in the graphs and use a modified gradient descent method to find appropriate parameters. Figure 1b shows the resulting activities of units in response to binocular horizontal (i) and vertical (ii) bars, and also the two equally likely explanations for rivalrous input (iii and iv). For rivalry,

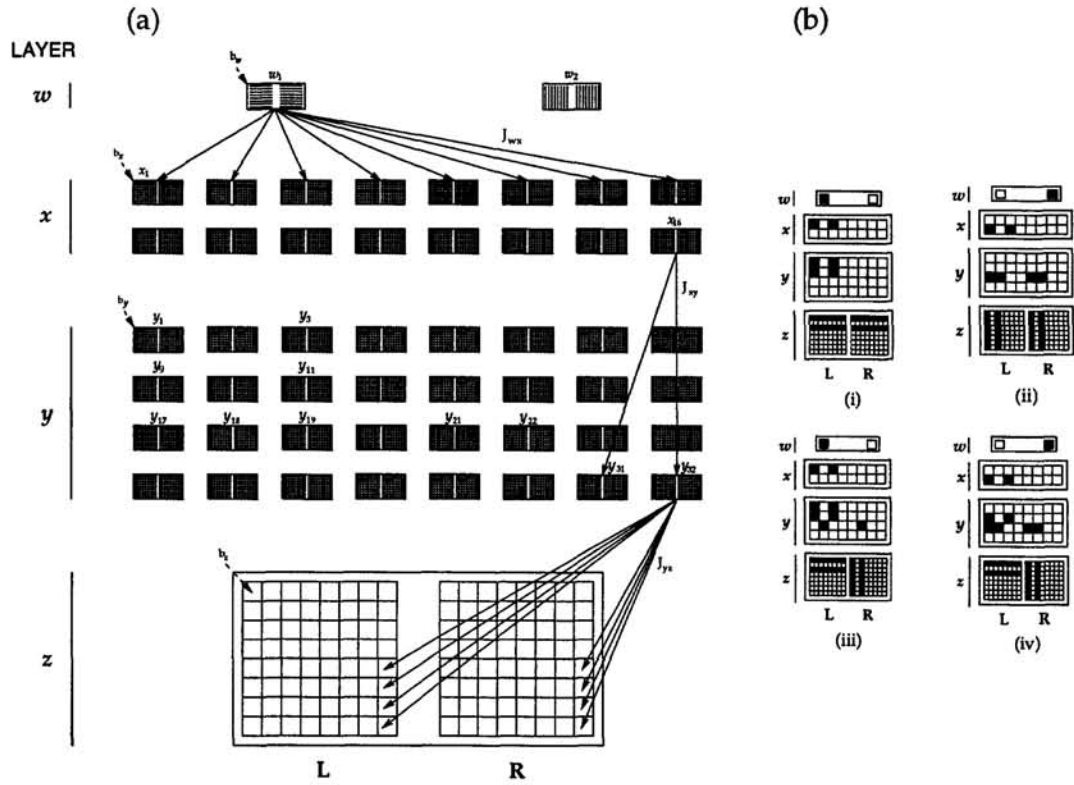

**Figure 1:** a) Hierarchical generative model for 8 × 8 bar patterns across the two eyes. Units are depicted by their net projective (generative) fields, and characteristic weights are shown. Even though the net projective field of $x_1$ is the top horizontal bar in both eyes, note that it generates this by increasing the probability that units $y_1$ and $y_9$ in the y layer will be active, not by having direct connections to the input z. Unit $w_1$ connects to $x_1, x_2, \ldots x_8$ through $J_{\mathbf{wx}} = 0.8$; $x_{16}$ connects to $y_{31}, y_{32}$ through $J_{\mathbf{xy}} = 1.0$ and $y_{32}$ connects to the bottom right half vertical bar through $J_{\mathbf{yz}} = 5.8$. Biases are $b_{\mathbf{w}} = -0.75, b_{\mathbf{x}} = -1.5, b_{\mathbf{y}} = -2.7$ and $b_{\mathbf{z}} = -3.3$. b) Recognition activity in the network for four different input patterns. The units are arranged in the same order as (a), and white and black squares imply activities for the units whose means are less than and greater than 0.5. (i) and (ii) represent normal binocular stimulation; (iii) and (iv) show the two alternative stable states during rivalrous stimulation, without the fatigue process.

there is direct competition in the top left hand quadrant of z, which is reflected in the competition between $y_1, y_3$ and $y_{17}, y_{21}$. However, the input regions (top right of L and bottom left of R) for which there is no competition, require the constant activity of explanations $y_9, y_{11}, y_{18}$ and $y_{22}$. Under the generative model, the coactivation of $y_1$ and $y_9$ *without* $x_1$ is quite unlikely ($\mathcal{P}[x_1 = 0|y_1 = 1, y_3 = 1] = 0.1$), which is why $x_1, x_3$ and also $w_1$ become active with $y_1$ and $y_3$.

Given just gradient descent for the rivalrous stimulus, the network would just find one of the two equally good (or rather bad) solutions in figure 1b(iii,iv). Alternation ensues when descent is augmented by a fatigue process:

$$\psi_1(t+1) = \psi_1(t) + \delta(-\nabla_{\psi_1}\mathcal{F}[\mu, \xi, \psi] + \alpha(\beta\psi_1(t)) - \psi_1'(t))$$
$$\psi_1'(t+1) = \psi_1'(t) + \delta(\psi_1(t) - \beta\psi_1'(t)),$$

where $\beta$ is a decay term. In all the simulations, $\alpha = 0.5, \beta = 0.1$ and $\delta = 0.01$.

We adopted various heuristics to simplify the process of using this rather cumbersome mean field model. First, fatigue is only implemented for the units in the y

layer, and the $\psi$ follow the equivalent of the dynamical equations above. Although adaptation processes can clearly occur at many levels in the system, and indeed have been used to try to diagnose the mechanisms of rivalry,[15] their exact form is not clear. Bialek & DeWeese[1] argue that the rate of a switching process should be adaptive to the expected rate of change of the associated signal on the basis of prior observations. This is clearly faster nearer to the input.

The second heuristic is that rather than perform gradient descent for the non-fatiguing units, the optimal values of $\mu$ and $\xi$ are calculated on each iteration by solving numerically equations such as

$$\nabla_{\xi_i} \mathcal{F}[\mu, \xi, \psi] = 0.$$

The dearth of connections in the network of figure 1a allows $\mu$ and $\xi$ to be calculated locally at each unit in an efficient manner. Whether this is reasonable depends on the time constants of settling in the mean field model with respect to the dynamics of switching, and, more particularly on the way that this deterministic model is made appropriately stochastic.

Figure 2a shows the resulting activities during rivalry of units at various levels of the hierarchy including the fatigue process. Broadly, the competing explanations in figure 1b(iii;iv), $ie$ horizontal and vertical percepts, alternate, and units without competing inputs, such as $y_9$, are much less modulated than the others, such as $y_1$. The activity of $y_9$ is slightly elevated when horizontal bars are dominant, based on top-down connections. The activities of the units higher up, such as $x_1$ and $w_1$, do not decrease to 0 during the suppression period for horizontal bars, leaving weak activity during suppression. Many of the modulating cells in monkeys were not completely silent during their periods of less activity.[16] Figure 2b shows that the hierarchical version of the model also behaves in accordance with experimental results on the effects of varying the input contrast,[17,10,22,16] which suggest that increasing the contrast in both eyes decreases the period of the oscillation ($ie$ increases the frequency), and increasing the contrast in just one eye decreases the suppression period for that eye much more than it increases its dominance period.

## 3   Discussion

Following Logothetis and his colleagues[19,16,18] (see also Grossberg[11]) we have suggested an account of rivalry based on competing top-down hierarchical explanations, and have shown how it models various experimental observations on rivalry. Neurons explain inputs in virtue of being capable of generating their activities through a top-down statistical generative model. Competition arises between higher-level explanations of overlapping active regions ($ie$ those involving contrast changes) of the input rather than between inputs themselves. Note that alternating the input between the two eyes would have no effect on this behaviour of the model, since $explanations$ are competing rather than $inputs$. Of course, the model is greatly simplified – for instance, it only has units that are not modulating with the percept in the earliest binocular layer (layer y), whereas in the monkeys, more than half the cells in V4 were unmodulated during rivalry.[16]

The model's accounts of the neurophysiological findings described in the introduction are: i) monocular cells will generally not be modulated if they are involved in

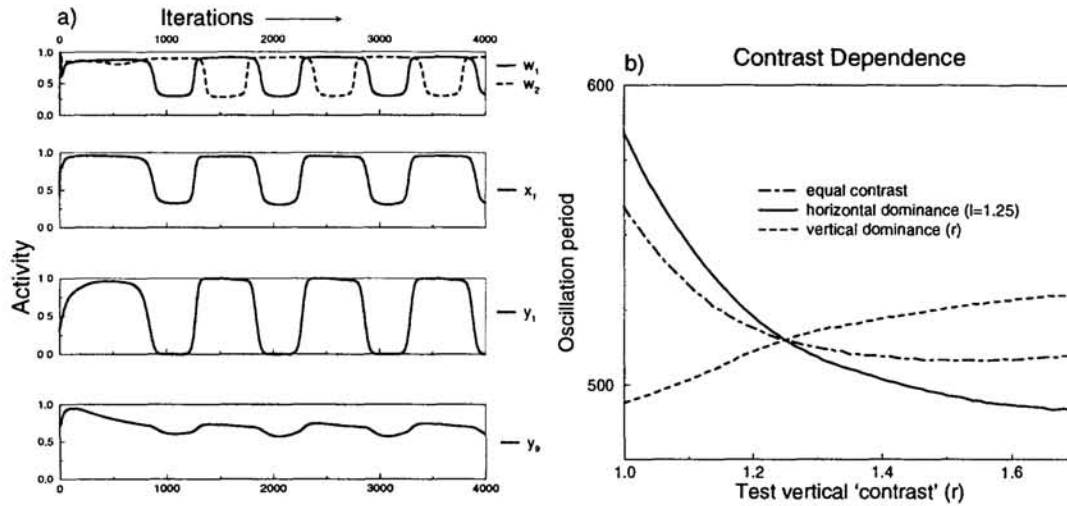

Figure 2: a) Mean activities of units at three levels of the hierarchy in response to rivalrous stimuli with input strengths $l = r = 1.75$. b) Contrast dependence of the oscillation periods for equal input strengths, and when $l = 1.25$ and $r$ is varied.

explaining local correlations in the input from a single eye. This model does not demonstrate this explicitly, but would if, for instance, each of the inputs $z_i$ actually consisted of two units, which are always on or off together. In this case one could get a compact explanation of the joint activities with a set of monocular units which would then not be modulated. ii) Units such as $y_9$ in the hierarchical model are binocular, are selective between the binocular version of the stimuli, and are barely modulated with the percept. iii) Units such as $y_1, x_1$ and $w_1$ are binocular, are selective between the stimuli, and are significantly modulated with the percept.

The final neurophysiological finding is to do with cells that fire when their preferred stimuli are suppressed, or fire selectively between the stimuli *only* during rivalry. There are no units in this model that are selective between the stimuli and are preferentially activated during *suppression* of their preferred stimuli. However, in a model with more complicated stimulus contingencies, they would emerge to account for the parts of the stimulus in the suppressed eye that are *not* accounted for by the explanation of the overlying parts of the dominant explanation, at least provided that this residual between the true monocular stimulus and the current explanation is sufficiently complex as to require explaining itself.

We would expect to find two sets of cells that are activated during the suppressed period by this residual, some of which will form part of the representation of the stimulus when presented binocularly and some of which will not. Those that do not (class A) will only even appear to be selective between the stimuli during rivalry, and will represent parts of the residual that are themselves explained by more overarching explanations for parts of the complete (binocularly presented) stimulus. This suggests the experimental test of presenting binocularly a putative form of the residual (*eg* dotted lines for competing horizontal and vertical gratings). We predict that these cells should be activated.

If there are cells that do participate in the binocular representation, then they will be selective, but will preferentially fire during suppression (class B). Certainly, the

residual will have a high correlation with the full suppressed pattern, and so a cell that is selective for part of the residual could have appropriate properties. However, why should such a cell not fire when the full, but currently suppressed, pattern is dominant? In monkeys,[16] there are fewer class B than class A cells (0 versus 3 of 33 cells in V1/2; 6 versus 8 of 68 cells in V4). Under the model, we account for these cells based on a competition between units that represent the residual and those that represent overlapping parts of the complete pattern. In binocular viewing, explanations are generally stronger than during rivalry. So even if both such units participate in representing a binocular stimulus, the cells representing the residual might not reach threshold during the dominance period. However, during suppression, they no longer suffer from competition, and so will be activated. The model's explanation for class B cells seems far less natural than that for class A cells. One experimental test would be to present the preferred pattern binocularly, reduce the contrast, and see if these cells are suppressed more strongly.

The overall model mechanistically has much in common with models which place the competition in rivalry at the level of binocular oriented cells rather than between monocular cells.[11,2] Indeed, the model is based on an explanation-driven account for normal binocular processing, so this is to be expected. The advantage of couching rivalry in terms of explanations is that this provides a natural way of accounting for top-down influences. In fact, one can hope to study top-down control through studying its effects on the behaviour of cells during rivalry.

The model suffers from various lacunæ. Foremost, it is necessary to model the stochasticity of switching between explanations.[9,17] The distributions of dominance times for both humans and monkeys is well characterised by a $\Gamma$ distribution (Lehky[14] argues that this is descriptive rather than normative), with strong independence between successive dominance periods. Our mean field recognition process is deterministic. The stochastic analogue would be some form of Markov chain Monte-Carlo method such as Gibbs sampling. However, it is not obvious how to incorporate the equivalent of fatigue in a computationally reasonable way. In any case, the nature of neuronal randomness is subject to significant debate at present. Note that the recognition model of a stochastic Helmholtz machine[7,6] would be unsuitable, since it is purely feedforward and does not integrate bottom-up and top-down information.

We have adopted a very simple mean field approach to recognition, giving up neurobiological plausibility for convenience. The determinism of the mean field model in any case rules it out as a complete explanation, but it does at least show clearly the nature of competition between explanations. The architecture of the model is also incomplete. The cortex is replete with what we would model as lateral connections between units within a single layer. We have constructed generative models in which there are no such direct connections, because they significantly complicate the mean field recognition method. It could be that these connections are important for the recognition process,[6] but modeling their effect would require representing them explicitly. This would also allow modeling of the apparent diffusive process by which patches of dominance spread and alter. In a complete model, it would also be necessary to account for competition between eyes in addition to competition between explanations.[24,8,3]

Another gap is some form of contrast gain control.[5] The model is quite sensitive to input contrast. This is obviously important for the effects shown in figures 2, however the range of contrasts over which it works should be larger. It would be particularly revealing to explore the effects of changing the contrast in some parts of images and examine the consequent effects on the spreading of dominance.

# References

[1] Bialek, W & DeWeese, M (1995). Random switching and optimal processing in the perception of ambiguous signals. *Physical Review Letters,* **74,** 3077-3080.

[2] Blake, R (1989). A neural theory of binocular rivalry. *Psychological Review,* **96,** 145-167.

[3] Blake, R & Fox, R (1974). Binocular rivalry suppression: Insensitive to spatial frequency and orientation change. *Vision Research,* **14,** 687-692.

[4] Blake, R, Westendorf, DH & Overton, R (1980). What is suppressed during binocular rivalry? *Perception,* **9,** 223-231.

[5] Carandini, M & Heeger, DJ (1994). Summation and division by neurons in primate visual cortex. *Science,* **264,** 1333-1336.

[6] Dayan, P & Hinton, GE (1996). Varieties of Helmholtz machine. *Neural Networks,* **9,** 1385-1403.

[7] Dayan, P, Hinton, GE, Neal, RM & Zemel, RS (1995). The Helmholtz machine. *Neural Computation,* **7,** 889-904.

[8] Fox, R & Check, R (1972). Independence between binocular rivalry suppression duration and magnitude of suppression. *Journal of Experimental Psychology,* **93,** 283-289.

[9] Fox, R & Herrmann, J (1967). Stochastic properties of binocular rivalry alternations. *Perception and Psychophysics,* **2,** 432-436.

[10] Fox, R & Rasche, F (1969). Binocular rivalry and reciprocal inhibition. *Perception and Psychophyics,* **5,** 215-217.

[11] Grossberg, S (1987). Cortical dynamics of three-dimensional form, color and brightness perception: 2. Binocular theory. *Perception & Psychphysics,* **41,** 117-158.

[12] Hinton, GE, Dayan, P, Frey, BJ & Neal, RM (1995). The wake-sleep algorithm for unsupervised neural networks. *Science,* **268,** 1158-1160.

[13] Jaakkola, T, Saul, LK & Jordan, MI (1996). Fast learning by bounding likelihoods in sigmoid type belief networks. *Advances in Neural Information Processing Systems, 8,* forthcoming.

[14] Lehky, SR (1988). An astable multivibrator model of binocular rivalry. *Perception,* **17,** 215-228.

[15] Lehky, SR & Blake, R (1991). Organization of binocular pathways: Modeling and data related to rivalry. *Neural Computation,* **3,** 44-53.

[16] Leopold, DA & Logothetis, NK (1996). Activity changes in early visual cortex reflect monkeys' percepts during binocular rivalry. *Nature,* **379,** 549-554.

[17] Levelt, WJM (1968). *On Binocular Rivalry.* The Hague, Paris: Mouton.

[18] Logothetis, NK, Leopold, DA & Sheinberg, DL (1996). What is rivalling during binocular rivalry. *Nature,* **380,** 621-624.

[19] Logothetis, NK & Schall, JD (1989). Neuronal correlates of subjective visual perception. *Science,,* **245,** 761-763.

[20] Matsuoka, K (1984). The dynamic model of binocular rivalry. *Biological Cybernetics,* **49,** 201-208.

[21] Mueller, TJ (1990). A physiological model of binocular rivalry. *Visual Neuroscience,* **4,** 63-73.

[22] Mueller, TJ & Blake, R (1989). A fresh look at the temporal dynamics of binocular rivalry. *Biological Cybernetics,* **61,** 223-232.

[23] Pearl, J (1988). *Probabilistic Reasoning in Intelligent Systems: Networks of Plausible Inference.* San Mateo, CA: Morgan Kaufmann.

[24] Wales, R & Fox, R (1970). Increment detection thresholds during binocular rivalry suppression. *Perception and Psychophysics,* **8,** 90-94.

[25] Wheatstone, C (1838). Contributions to the theory of vision. I: On some remarkable and hitherto unobserved phenomena of binocular vision. *Philosophical Transactions of the Royal Society of London,* **128,** 371-394.

[26] Wolfe, JM (1986). Stereopsis and binocular rivalry. *Psychological Review,* **93,** 269-282.
